# Beat Tracking the Graphical Model Way

**Dustin Lang**    **Nando de Freitas**
Department of Computer Science
University of British Columbia
Vancouver, BC
`{dalang, nando}@cs.ubc.ca`

## Abstract

We present a graphical model for beat tracking in recorded music. Using a probabilistic graphical model allows us to incorporate local information and global smoothness constraints in a principled manner. We evaluate our model on a set of varied and difficult examples, and achieve impressive results. By using a fast dual-tree algorithm for graphical model inference, our system runs in less time than the duration of the music being processed.

## 1 Introduction

This paper describes our approach to the beat tracking problem. Dixon describes *beats* as follows: "much music has as its rhythmic basis a series of pulses, spaced approximately equally in time, relative to which the timing of all musical events can be described. This phenomenon is called the *beat*, and the individual pulses are also called beats"[1]. Given a piece of recorded music (an MP3 file, for example), we wish to produce a set of beats that correspond to the beats perceived by human listeners.

The set of beats of a song can be characterised by the trajectories through time of the *tempo* and *phase offset*. Tempo is typically measured in beats per minute (BPM), and describes the frequency of beats. The phase offset determines the time offset of the beat. When tapping a foot in time to music, tempo is the rate of foot tapping and phase offset is the time at which the tap occurs.

The beat tracking problem, in its general form, is quite difficult. Music is often ambiguous; different human listeners can perceive the beat differently. There are often several beat tracks that could be considered correct. Human perception of the beat is influenced both by 'local' and contextual information; the beat can continue through several seconds of silence in the middle of a song.

We see the beat tracking problem as not only an interesting problem in its own right, but as one aspect of the larger problem of machine analysis of music. Given beat tracks for a number of songs, we could extract descriptions of the rhythm and use these features for clustering or searching in music collections. We could also use the rhythm information to do structural analysis of songs - for example, to find repeating sections. In addition, we note that beat tracking produces a description of the time scale of a song; knowledge of the tempo of a song would be one way to achieve time-invariance in a symbolic description. Finally, we note that beat tracking tells us where the important parts of a song are; the

beats (and major divisions of the beats) are good sampling points for other music-analysis problems such as note detection.

## 2 Related Work

Many researchers have investigated the beat tracking problem; we present only a brief overview here. Scheirer [2] presents a system, based on psychoacoustical observations, in which a bank of resonators compete to explain the processed audio input. The system is tested on a difficult set of examples, and considerable success is reported. The most common problem is a lack of global consistency in the results - the system switches between locally optimal solutions.

Goto [3] has described several systems for beat tracking. He takes a very pragmatic view of the problem, and introduces a number of assumptions that allow good results in a limited domain - pop music in 4/4 time with roughly constant tempo, where bass or snare drums keep the beat according to drum patterns known *a priori*, or where chord changes occur at particular times within the measure.

Cemgil and Kappen [4] phrase the beat tracking problem in probabilistic terms, and we adapt their model as our local observation model. They use MIDI-like (event-based) input rather than audio, so the results are not easily comparable to our system.

## 3 Graphical Model

In formulating our model for beat tracking, we assume that the tempo is nearly constant over short periods of time, and usually varies smoothly. We expect the phase to be continuous. This allows us to use the simple graphical model shown in Figure 1. We break the song into a set of *frames* of two seconds; each frame is a node in the graphical model. We expect the tempo to be constant within each frame, and the tempo and phase offset parameters to vary smoothly between frames.

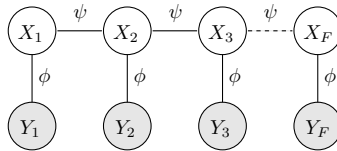

Figure 1: Our graphical model for beat tracking. The *hidden state* $\boldsymbol{X}$ is composed of the state variables tempo and phase offset. The *observations* $\boldsymbol{Y}$ are the features extracted by our audio signal processing. The potential function $\phi$ describes the compatibility of the observations with the state, while the potential function $\psi$ describes the smoothness between neighbouring states.

In this undirected probabilistic graphical model, the potential function $\phi$ describes the compatibility of the state variables $\boldsymbol{X} = \{T, P\}$ composed of tempo $T$ and phase offset $P$ with the local observations $\boldsymbol{Y}$. The potential function $\psi$ describes the smoothness constraints between frames. The observation $\boldsymbol{Y}$ comes from processing the audio signal, which is described in Section 5. The $\phi$ function comes from domain knowledge and is described in Section 4. This model allows us to trade off local fit and global smoothness in a principled manner. By using an undirected model, we allow contextual information to flow both forward and backward in time.

In such models, belief propagation (BP) [5] allows us to compute the marginal probabilities of the state variables in each frame. Alternatively, maximum belief propagation (max-BP)

allows a joint maximum *a posteriori* (MAP) set of state variables to be determined. That is, given a song, we generate the observations $Y_i$, $i = 1 \ldots F$, (where $F$ is the number of frames in the song) and seek a set of states $X_i$ that maximize the joint product

$$P(\boldsymbol{X}, \boldsymbol{Y}) = \frac{1}{Z} \prod_{i=1}^{F} \phi(\boldsymbol{Y}_i, \boldsymbol{X}_i) \prod_{i=1}^{F-1} \psi(\boldsymbol{X}_i, \boldsymbol{X}_{i+1}) \quad .$$

Our smoothness function $\psi$ is the product of tempo and phase smoothness components $\psi_T$ and $\psi_P$. For the tempo component, we use a Gaussian on the log of tempo. For the phase offset component, we want the phases to agree at a particular point in time: the boundary between the two frames (nodes), $t_b$. We find the phase $\theta$ of $t_b$ predicted by the parameters in each frame, and place a Gaussian prior on the distance between points on the unit circle with these phases:

$$\begin{aligned}
\psi(\boldsymbol{X}_1, \boldsymbol{X}_2 \mid t_b) &= \psi_T(T_1, T_2) \, \psi_P(T_1, P_1, T_2, P_2 \mid t_b) \\
&= \mathcal{N}(\log T_1 - \log T_2, \sigma_T^2) \, \mathcal{N}((\cos\theta_1 - \cos\theta_2, \sin\theta_1 - \sin\theta_2), \sigma_P^2)
\end{aligned}$$

where $\theta_i = 2\pi T_i t_b - P_i$ and $\mathcal{N}(x, \sigma^2)$ is a zero-mean Gaussian with variance $\sigma^2$. We set $\sigma_T = 0.1$ and $\sigma_P = 0.1\pi$. The qualitative results seem to be fairly stable as a function of these smoothness parameters.

## 4 Domain Knowledge

In this section, we describe the derivation of our local potential function (also known as the observation model) $\phi(\boldsymbol{Y}_i, \boldsymbol{X}_i)$.

Our model is an adaptation of the work of [4], which was developed for use with MIDI input. Their model is designed so that it "prefers simpler [musical] notations". The beat is divided into a fixed number of bins (some power of two), and each note is assigned to the nearest bin. The probability of observing a note at a coarse subdivision of the beat is greater than at a finer subdivision. More precisely, a note that is quantized to the bin at beat number $k$ has probability $p(k) \propto \exp(-\lambda \, d(k))$, where $d(k)$ is the number of digits in the binary representation of the number $k \bmod 1$.

Since we use recorded music rather than MIDI, we must perform signal processing to extract features from the raw data. This process produces a signal that has considerably more uncertainty than the discrete events of MIDI data, so we adjust the model. We add the constraint that features should be observed near *some* quantization point, which we express by centering a Gaussian around each of the quantization points. The variance of this Gaussian, $\sigma_Q^2$ is in units of beats, so we arrive at the periodic template function $b(t)$, shown in Figure 2. We have set the number of bins to 8, $\lambda$ to one, and $\sigma_Q = 0.025$.

The template function $b(t)$ expresses our belief about the distribution of musical events within the beat. By shifting and scaling $b(t)$, we can describe the expected distribution of notes in time for different tempos and phase offsets:

$$b(t \mid T, P) = b\left(Tt - \frac{P}{2\pi}\right) \quad .$$

Our signal processing (described below) yields a discrete set of events that are meant to correspond to musical events. Events occur at a particular time $t$ and have a 'strength' or 'energy' $E$. Given a set of discrete events $\boldsymbol{Y} = \{t_i, E_i\}$, $i = 1 \ldots M$, and state variables $\boldsymbol{X} = \{T, P\}$, we take the probability that the events were drawn from the expected distribution $b(t \mid T, P)$:

$$\phi(\boldsymbol{Y}, \boldsymbol{X}) = \phi(\{t, E\}, \{T, P\}) = \prod_{i=1}^{M} b(t_i \mid T, P)^{E_i} \quad .$$

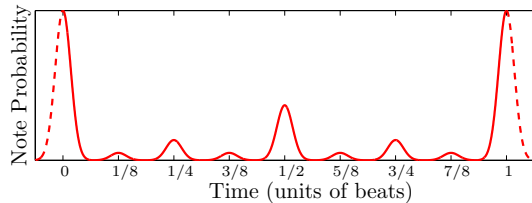

Figure 2: One period of our template function $b(t)$, which gives the expected distribution of notes within a beat. Given tempo and phase offset values, we stretch and shift this function to get the expected distribution of notes in time.

This is a multinomial probability function in the continuous limit (as the bin size becomes zero). Note that $\phi$ is a positive, unnormalized potential function.

## 5 Signal Processing

Our signal processing stage is meant to extract features that approximate musical events (drum beats, piano notes, guitar strums, etc.) from the raw audio signal. As discussed above, we produce a set of events composed of time and 'strength' values, where the strength describes our certainty that an event occurred. We assume that musical events are characterised by brief, rapid increases in energy in the audio signal. This is certainly the case for percussive instruments such as drums and piano, and will often be the case for string and woodwind instruments and for voices. This assumption breaks for sounds that fade in smoothly rather than 'spikily'.

We begin by taking the short-time Fourier transform (STFT) of the signal: we slide a 50 millisecond Hann window over the signal in steps of 10 milliseconds, take the Fourier transform, and extract the energy spectrum. Following a suggestion by [2], we pass the energy spectrum through a bank of five filters that sum the energy in different portions of the spectrum. We take the logarithm of the summed energies to get a 'loudness' signal. Next, we convolve each of the five resulting energy signals with a filter that detects positive-going edges. This can be considered a 'loudness gain' signal. Finally, we find the maxima within 50 ms neighbourhoods. The result is a set of points that describe the energy gain signal in each band, with emphasis on the maxima. These are the features $\boldsymbol{Y}$ that we use in our local probability model $\phi$.

## 6 Fast Inference

To find a maximum *a posteriori* (MAP) set of state variables that best explain a set of observations, we need to optimize a $2F$-dimensional, continuous, non-linear, non-Gaussian function that has many local extrema. $F$ is the number of frames in the song, so is on the order of the length of the song in seconds - typically in the hundreds. This is clearly difficult. We present two approximation strategies. In the first strategy, we convert the continuous state space into a uniform discrete grid and run discrete belief propagation. In the second strategy, we run a particle filter in the forward direction, then use the particles as 'grid' points and run discrete belief propagation as per [6].

Since the landscape we are optimizing has many local maxima, we must use a fine discretization grid (for the first strategy) or a large number of particles (for the second strategy). The message-passing stage in discrete belief propagation takes $O(N^2)$ if performed naively, where $N$ is the number of discretized states (or particles) per frame. We use a dual-tree recursion strategy as proposed in [7] and extended to maximum *a posteriori* inference

in [8]. With this approach, the computation becomes feasible.

As an aside, we note that if we wish to compute the smoothed marginal probabilities rather than the MAP set of parameters, then we can use standard discrete belief propagation or particle smoothing. In both cases, the naive cost in $O(N^2)$, but by using the Fast Gauss Transform[9] the cost becomes $O(N)$. This is possible because our smoothness potential $\psi$ is a low-dimensional Gaussian.

For the results presented here, we discretize the state space into $N_T = 90$ tempo values and $N_P = 50$ phase offset values for the belief propagation version. We distribute the tempo values uniformly on a log scale between 40 and 150 BPM, and distribute the phase offsets uniformly. For the particle filter version, we use $N_T \times N_P = 4500$ particles. With these values, our *Matlab* and *C* implementation runs at faster than real time (the duration of the song) on a standard desktop computer.

## 7 Results

A standard corpus of labelled ground truth data for the beat-tracking problem does not exist. Therefore, we labelled a relatively small number of songs for evaluation of our algorithm, by listening to the songs and pressing a key at each perceived beat. We sought out examples that we thought would be difficult, and we attempted to avoid the methods of [10]. Ideally, we would have several human listeners label each song, since this would help to capture the ambiguity inherent in the problem. However, this would be quite time-consuming.

One can imagine several methods for speeding up the process of generating ground truth labellings and of cleaning up the noisy results generated by humans. For example, a human labelling of a short segment of the song could be automatically extrapolated to the remainder of the song, using energy spikes in the audio signal to fine-tune the placement of beats. However, by generating ground truth using assumptions similar to those embodied in the models we intend to test, we risk invalidating the results. We instead opted to use 'raw' human-labelled songs.

There is no standard evaluation metric for beat tracking. We use the $\rho$ function presented by Cemgil *et al* [11] and used by Dixon [1] in his analysis:

$$\rho(S,T) = \frac{100}{(N_S + N_T)/2} \sum_{i=1}^{N_S} \max_{j \in T} \exp\left\{-\frac{(S_i - T_j)^2}{2\sigma^2}\right\}$$

where $S$ and $T$ are the ground-truth and proposed beat times, and $\sigma$ is set to 40 milliseconds. A $\rho$ value near 100 means that each predicted beat in close to a true beat, while a value near zero means that each predicted beat is far from a true beat.

We have focused on finding a globally-optimum beat track rather than precisely locating each beat. We could likely improve the $\rho$ values of our results by fine-tuning each predicted beat, for example by finding nearby energy peaks, though we have not done this in the results presented here.

Table 1 shows a summary of our results. Note the wide range of genres and the choice of songs with features that we thought would make beat tracking difficult. This includes all our results (not just the ones that look good).

The first columns list the name of the song and the reason we included it. The third column lists the qualitative performance of the fixed grid version: *double* means our algorithm produced a beat track twice as fast as ground truth, *half* means we tracked at half speed, and *sync* means we produced a syncopated ($\pi$ phase error) beat track. A blank entry means our algorithm produced the correct beat track. A star ($\star$) means that our result incorrectly switches phase or tempo. The $\rho$ values are after compensating for the qualitative error (if any). The fifth column shows a histogram of the absolute phase error (0 to $\pi$); this is also

| Song | Comment | BP Perf. | BP $\rho$ | Phase Err | PF Perf. | PF $\rho$ | Phase Err |
|---|---|---|---|---|---|---|---|
| Glenn Gould / Bach Goldberg Var'ns 1982 / Var'n 1 | Classical piano | | 88 | | | 86 | |
| Jeno Jandó / Bach WTC / Fuga 2 (C Minor) | Piano; rubato at end | | 77 | | | 77 | |
| Kronos Quartet / Caravan / Aaj Ki Raat | Modern string quartet | | 75 | | | 71 | |
| Maurice Ravel / Piano Concertos / G Major - Presto | Classical orchestra | ⋆ sync | 44 | | ⋆ sync | 50 | |
| Miles Davis / Kind Of Blue / So What (edit) | Jazz instrumental | half | 61 | | half | 59 | |
| Miles Davis / Kind Of Blue / Blue In Green | Jazz instrumental | | 57 | | | 59 | |
| Holly Cole / Temptation / Jersey Girl | Jazz vocal | | 78 | | | 77 | |
| Don Ross / Passion Session / Michael Michael Michael | Solo guitar | ⋆ threehalf | 40 | | ⋆ threehalf | 42 | |
| Don Ross / Huron Street / Luci Watusi | Solo guitar | | 70 | | | 69 | |
| Tracy Chapman / For You | Guitar and voice | double | 59 | | double | 61 | |
| Ben Harper / Fight For Your Mind / Oppression | Acoustic | | 70 | | | 68 | |
| Great Big Sea / Up / Chemical Worker's Song | Newfoundland folk | | 79 | | | 78 | |
| Buena Vista Social Club / Chan Chan | Cuban | | 72 | | | 72 | |
| Beatles / 1967-1970 / Lucy In The Sky With Diamonds | Changes time signature | ⋆ | 42 | | ⋆ | 41 | |
| U2 / Joshua Tree / Where The Streets Have No Name (edit) | Rock | | 82 | | | 82 | |
| Cake / Fashion Nugget / I Will Survive | Rock | sync | 81 | | sync | 80 | |
| Sublime / Second-Hand Smoke / Thanx Dub (excerpt) | Reggae | | 79 | | | 79 | |
| Rancid / ... And Out Come The Wolves / Old Friend | Punk | half | 82 | | half | 79 | |
| Green Day / Dookie / When I Come Around | Pop-punk | | 75 | | | 74 | |
| Tortoise / TNT / A Simple Way To Go Faster Than Light... | Organic electronica | double | 79 | | double | 79 | |
| Pole / 2 / Stadt | Ambient electronica | | 71 | | | 71 | |
| Underworld / A Hundred Days Off / MoMove | Electronica | | 79 | | | 79 | |
| Ravi Shankar / The Sounds Of India / Bhimpalsi (edit) | Solo sitar | | 71 | | | 67 | |
| Pitamaha: Music From Bali / Puri Bagus, Bamboo (excerpt) | Indonesian gamelan | | 86 | | sync | 89 | |
| Gamelan Sekar Jaya / Byomantara (excerpt) | Indonesian gamelan | | 89 | | | 88 | |

Table 1: The songs used in our evaluation. See the text for explanation.

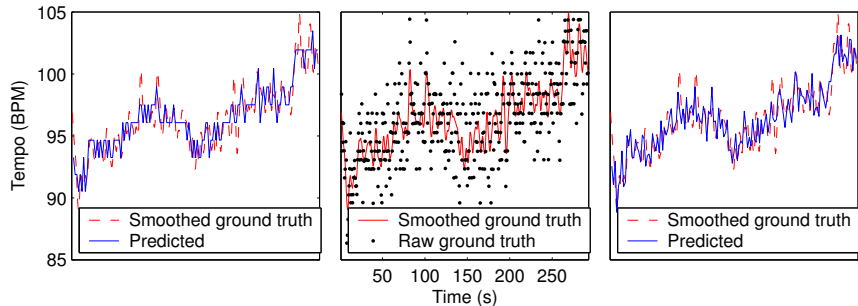

Figure 3: Tempo tracks for *Cake / I Will Survive*. Center: 'raw' ground-truth tempo (instantaneous tempo estimate based on the time between adjacent beats) and smoothed ground truth (by averaging). Left: fixed-grid version result. Right: particle filter result.

after correcting for qualitative error. The remaining columns contain the same items for the particle filter version.

Out of 25 examples, the fixed grid version produces the correct answer in 17 cases, tracks at double speed in two cases, half speed in two cases, syncopated in one case, and in three cases produces a track that (incorrectly) switches tempo or phase. The particle filter version produces 16 correct answers, two double-speed, two half-speed, two syncopated, and the same three 'switching' tracks.

An example of a successful tempo track is shown in Figure 3.

The result for *Lucy In The Sky With Diamonds* (one of the 'switching' results) is worth examination. The song switches time signature between 3/4 and 4/4 time a total of five times; see Figure 4. Our results follow the time signature change the first three times. On the fourth change (from 4/4 to 3/4), it tracks at 2/3 the ground truth rate instead. We note an interesting effect when we examine the final message that is passed during belief propagation. This message tells us the maximum probability of a sequence that ends with each state. The global maximum corresponds to the beat track shown in the left plot. The local maximum near 50 BPM corresponds to an alternate solution in which, rather than tracking the quarter notes, we produce one beat per measure; this track is quite plausible. Indeed, the 'true' track is difficult for human listeners. Note also that there is also a local maximum near 100 BPM but phase-shifted a half beat. This is the solution in which the beats are syncopated from the true result.

## 8   Conclusions and Further Work

We present a graphical model for beat tracking and evaluate it on a set of varied and difficult examples. We achieve good results that are comparable with those reported by other researchers, although direct comparisons are impossible without a shared data set.

There are several advantages to formulating the problem in a probabilistic setting. The beat tracking problem has inherent ambiguity and multiple interpretations are often plausible. With a probabilistic model, we can produce several candidate solutions with different probabilities. This is particularly useful for situations in which beat tracking is one element in a larger machine listening application. Probabilistic graphical models allow flexible and powerful handling of uncertainty, and allow local and contextual information to interact in a principled manner. Additional domain knowledge and constraints can be added in a clean and principled way. The adoption of an efficient dual tree recursion for graphical models

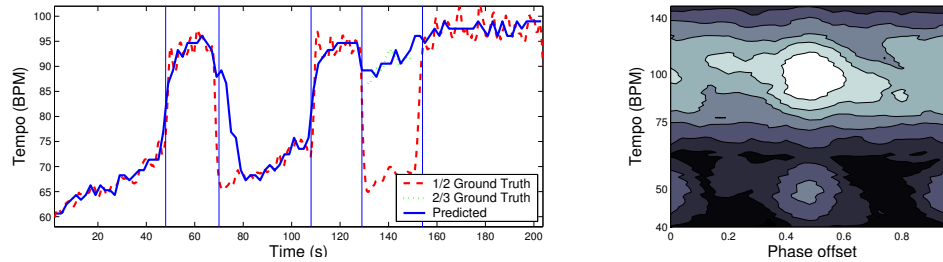

Figure 4: Left: Tempo tracks for *Lucy In The Sky With Diamonds*. The vertical lines mark times at which the time signature changes between 3/4 and 4/4. Right: the last max-message computed during belief propagation. Bright means high probability. The global maximum corresponds to the tempo track shown. Note the local maximum around 50 BPM, which corresponds to an alternate feasible result. See the text for discussion.

[7, 8] enables us to carry out inference in real time.

We would like to investigate several modifications of our model and inference methods. Longer-range tempo smoothness constraints as suggested by [11] could be useful. The extraction of MAP sets of parameters for several qualitatively different solutions would help to express the ambiguity of the problem. The particle filter could also be changed. At present, we first perform a full particle filtering sweep and then run max-BP. Taking into account the quality of the partial MAP solutions during particle filtering might allow superior results by directing more particles toward regions of the state space that are likely to contain the final MAP solution. Since we know that our probability terrain is multi-modal, a mixture particle filter would be useful [12].

# References

[1] S Dixon. An empirical comparison of tempo trackers. Technical Report TR-2001-21, Austrian Research Institute for Artificial Intelligence, Vienna, Austria, 2001.

[2] E D Scheirer. Tempo and beat analysis of acoustic musical signals. *J. Acoust. Soc. Am.*, 103(1):588–601, Jan 1998.

[3] M Goto. An audio-based real-time beat tracking system for music with or without drum-sounds. *Journal of New Music Research*, 30(2):159–171, 2001.

[4] A T Cemgil and H J Kappen. Monte Carlo methods for tempo tracking and rhythm quantization. *Journal of Artificial Intelligence Research*, 18(1):45–81, 2003.

[5] J Pearl. *Probabilistic reasoning in intelligent systems: networks of plausible inference*. Morgan-Kaufmann, 1988.

[6] S J Godsill, A Doucet, and M West. Maximum a posteriori sequence estimation using Monte Carlo particle filters. *Ann. Inst. Stat. Math.*, 53(1):82–96, March 2001.

[7] A G Gray and A W Moore. 'N-Body' problems in statistical learning. In *Advances in Neural Information Processing Systems 4*, pages 521–527, 2000.

[8] M Klaas, D Lang, and N de Freitas. Fast maximum *a posteriori* inference in monte carlo state space. In *AI-STATS*, 2005.

[9] L Greengard and J Strain. The fast Gauss transform. *SIAM Journal of Scientific Statistical Computing*, 12(1):79–94, 1991.

[10] D LaLoudouana and M B Tarare. Data set selection. Presented at NIPS Workshop, 2002.

[11] A T Cemgil, B Kappen, P Desain, and H Honing. On tempo tracking: Tempogram representation and Kalman filtering. *Journal of New Music Research*, 28(4):259–273, 2001.

[12] J Vermaak, A Doucet, and Patrick Pérez. Maintaining multi-modality through mixture tracking. In *ICCV*, 2003.
